# On the Generalization Ability of
# Online *Strongly* Convex Programming Algorithms

**Sham M. Kakade**
TTI Chicago
Chicago, IL 60637
sham@tti-c.org

**Ambuj Tewari**
TTI Chicago
Chicago, IL 60637
tewari@tti-c.org

## Abstract

This paper examines the generalization properties of online convex programming algorithms when the loss function is Lipschitz and strongly convex. Our main result is a sharp bound, that holds with high probability, on the excess risk of the output of an online algorithm in terms of the average regret. This allows one to use recent algorithms with logarithmic cumulative regret guarantees to achieve fast convergence rates for the excess risk with high probability. As a corollary, we characterize the convergence rate of PEGASOS (with high probability), a recently proposed method for solving the SVM optimization problem.

## 1 Introduction

Online regret minimizing algorithms provide some of the most successful algorithms for many machine learning problems, both in terms of the speed of optimization and the quality of generalization. Notable examples include efficient learning algorithms for structured prediction [Collins, 2002] (an algorithm now widely used) and for ranking problems [Crammer et al., 2006] (providing competitive results with a fast implementation).

Online convex optimization is a sequential paradigm in which at each round, the learner predicts a vector $\mathbf{w}_t \in S \subset \mathbb{R}^n$, nature responds with a convex loss function, $\ell_t$, and the learner suffers loss $\ell_t(\mathbf{w}_t)$. In this setting, the goal of the learner is to minimize the regret:

$$\sum_{t=1}^{T} \ell_t(\mathbf{w}_t) - \min_{\mathbf{w} \in S} \sum_{t=1}^{T} \ell_t(\mathbf{w})$$

which is the difference between his cumulative loss and the cumulative loss of the optimal fixed vector.

Typically, these algorithms are used to train a learning algorithm incrementally, by sequentially feeding the algorithm a data sequence, $(X_1, Y_1), \ldots, (X_T, Y_T)$ (generated in an i.i.d. manner). In essence, the loss function used in the above paradigm at time $t$ is $\ell(\mathbf{w}; (X_t, Y_t))$, and this leads to a guaranteed bound on the regret:

$$\text{Reg}_T = \sum_{t=1}^{T} \ell(\mathbf{w}_t; (X_t, Y_t)) - \min_{\mathbf{w} \in S} \sum_{t=1}^{T} \ell(\mathbf{w}; (X_t, Y_t))$$

However, in the batch setting, we are typically interested in finding a parameter $\widehat{\mathbf{w}}$ with good generalization ability, i.e. we would like:

$$R(\widehat{\mathbf{w}}) - \min_{\mathbf{w} \in S} R(\mathbf{w})$$

to be small, where $R(\mathbf{w}) := \mathbb{E}\left[\ell(\mathbf{w}; (X, Y))\right]$ is the *risk*.

Intuitively, it seems plausible that low regret on an i.i.d. sequence, should imply good generalization performance. In fact, for most of the empirically successful online algorithms, we have a set of techniques to understand the generalization performance of these algorithms on new data via 'online to batch' conversions — the conversions relate the regret of the algorithm (on past data) to the generalization performance (on future data). These include cases which are tailored to general convex functions [Cesa-Bianchi et al., 2004] (whose regret is $O(\sqrt{T})$) and mistake bound settings [Cesa-Bianchi and Gentile, 2008] (where the the regret could be $O(1)$ under separability assumptions). In these conversions, we typically choose $\widehat{\mathbf{w}}$ to be the average of the $\mathbf{w}_t$ produced by our online algorithm.

Recently, there has been a growing body of work providing online algorithms for *strongly* convex loss functions (i.e. $\ell_t$ is strongly convex), with regret guarantees that are merely $O(\ln T)$. Such algorithms have the potential to be highly applicable since many machine learning optimization problems are in fact strongly convex — either with strongly convex loss functions (e.g. log loss, square loss) or, indirectly, via strongly convex regularizers (e.g. $L_2$ or $KL$ based regularization). Note that in the latter case, the loss function itself may only be just convex but a strongly convex regularizer effectively makes this a strongly convex optimization problem; e.g. the SVM optimization problem uses the hinge loss with $L_2$ regularization. In fact, for this case, the PEGASOS algorithm of Shalev-Shwartz et al. [2007] — based on the online strongly convex programming algorithm of Hazan et al. [2006] — is a state-of-the-art SVM solver. Also, Ratliff et al. [2007] provide a similar subgradient method for max-margin based structured prediction, which also has favorable empirical performance.

The aim of this paper is to examine the generalization properties of online convex programming algorithms when the loss function is strongly convex (where strong convexity can be defined in a general sense, with respect to some arbitrary norm $||\cdot||$). Suppose we have an online algorithm which has some guaranteed cumulative regret bound $\mathrm{Reg}_T$ (e.g. say $\mathrm{Reg}_T \leq \ln T$ with $T$ samples). Then a corollary of our main result shows that with probability greater than $1 - \delta \ln T$, we obtain a parameter $\widehat{\mathbf{w}}$ from our online algorithm such that:

$$R(\widehat{\mathbf{w}}) - \min_{\mathbf{w}} R(\mathbf{w}) \leq \frac{\mathrm{Reg}_T}{T} + O\left(\frac{\sqrt{\mathrm{Reg}_T \ln \frac{1}{\delta}}}{T} + \frac{\ln \frac{1}{\delta}}{T}\right) .$$

Here, the constants hidden in the $O$-notation are determined by the Lipschitz constant and the strong convexity parameter of the loss $\ell$. Importantly, note that the correction term is of lower order than the regret — if the regret is $\ln T$ then the additional penalty is $O(\frac{\sqrt{\ln T}}{T})$. If one naively uses the Hoeffding-Azuma methods in Cesa-Bianchi et al. [2004], one would obtain a significantly worse penalty of $O(1/\sqrt{T})$.

This result solves an open problem in Shalev-Shwartz et al. [2007], which was on characterizing the convergence rate of the PEGASOS algorithm, with high probability. PEGASOS is an online strongly convex programming algorithm for the SVM objective function — it repeatedly (and randomly) subsamples the training set in order to minimize the empirical SVM objective function. A corollary to this work essentially shows the convergence rate of PEGASOS (as a randomized optimization algorithm) is concentrated rather sharply.

Ratliff et al. [2007] also provide an online algorithm (based on Hazan et al. [2006]) for max-margin based structured prediction. Our results are also directly applicable in providing a sharper concentration result in their setting (In particular, see the regret bound in Equation 15, for which our results can be applied to).

This paper continues the line of research initiated by several researchers [Littlestone, 1989, Cesa-Bianchi et al., 2004, Zhang, 2005, Cesa-Bianchi and Gentile, 2008] which looks at how to convert online algorithms into batch algorithms with provable guarantees. Cesa-Bianchi and Gentile [2008] prove faster rates in the case when the *cumulative loss* of the online algorithm is small. Here, we are interested in the case where the *cumulative regret* is small. The work of Zhang [2005] is closest to ours. Zhang [2005] explicitly goes via the exponential moment method to derive sharper concentration results. In particular, for the regression problem with squared loss, Zhang [2005] gives a result similar to ours (see Theorem 8 therein). The present work can also be seen as generalizing his result to the case where we have strong convexity with respect to a general norm. Coupled with

recent advances in low regret algorithms in this setting, we are able to provide a result that holds more generally.

Our key technical tool is a probabilistic inequality due to Freedman [Freedman, 1975]. This, combined with a variance bound (Lemma 1) that follows from our assumptions about the loss function, allows us to derive our main result (Theorem 2). We then apply it to statistical learning with bounded loss, and to PEGASOS in Section 4.

## 2   Setting

Fix a compact convex subset $S$ of some space equipped with a norm $\|\cdot\|$. Let $\|\cdot\|_*$ be the dual norm defined by $\|\mathbf{v}\|_* := \sup_{\mathbf{w}\,:\,\|\mathbf{w}\|\leq 1} \mathbf{v}\cdot\mathbf{w}$. Let $Z$ be a random variable taking values in some space $\mathcal{Z}$. Our goal is to minimize $F(\mathbf{w}) := \mathbb{E}\left[f(\mathbf{w};Z)\right]$ over $\mathbf{w}\in S$. Here, $f : S\times\mathcal{Z}\to[0,B]$ is some function satisfying the following assumption.

**Assumption LIST.** *(LIpschitz and STrongly convex assumption)* For all $z\in\mathcal{Z}$, the function $f_z(\mathbf{w}) = f(\mathbf{w};z)$ is convex in $\mathbf{w}$ and satisfies:

1. $f_z$ has Lipschitz constant $L$ w.r.t. to the norm $\|\cdot\|$, i.e. $\forall\mathbf{w}\in S, \forall\boldsymbol{\lambda}\in\partial f_z(\mathbf{w})$ ($\partial f_z$ denotes the subdifferential of $f_z$), $\|\boldsymbol{\lambda}\|_* \leq L$. Note that this assumption implies $\forall\mathbf{w},\mathbf{w}'\in S$, $|f_z(\mathbf{w}) - f_z(\mathbf{w}')| \leq L\|\mathbf{w} - \mathbf{w}'\|$.

2. $f_z$ is *$\nu$-strongly convex* w.r.t. $\|\cdot\|$, i.e. $\forall\theta\in[0,1], \forall\mathbf{w},\mathbf{w}'\in S$,
$$f_z(\theta\mathbf{w} + (1-\theta)\mathbf{w}') \leq \theta f_z(\mathbf{w}) + (1-\theta)f_z(\mathbf{w}') - \frac{\nu}{2}\theta(1-\theta)\|\mathbf{w} - \mathbf{w}'\|^2 .$$

Denote the minimizer of $F$ by $\mathbf{w}^\star$, $\mathbf{w}^\star := \arg\min_{\mathbf{w}\in S}F(\mathbf{w})$. We consider an online setting in which independent (but not necessarily identically distributed) random variables $Z_1,\ldots,Z_T$ become available to us in that order. These have the property that
$$\forall t, \forall\mathbf{w}\in S,\ \mathbb{E}\left[f(\mathbf{w};Z_t)\right] = F(\mathbf{w}) .$$

Now consider an algorithm that starts out with some $\mathbf{w}_1$ and at time $t$, having seen $Z_t$, updates the parameter $\mathbf{w}_t$ to $\mathbf{w}_{t+1}$. Let $\mathbb{E}_{t-1}\left[\cdot\right]$ denote conditional expectation w.r.t. $Z_1,\ldots,Z_{t-1}$. Note that $\mathbf{w}_t$ is measurable w.r.t. $Z_1,\ldots,Z_{t-1}$ and hence $\mathbb{E}_{t-1}\left[f(\mathbf{w}_t;Z_t)\right] = F(\mathbf{w}_t)$.

Define the statistics,
$$\mathrm{Reg}_T := \sum_{t=1}^T f(\mathbf{w}_t;Z_t) - \min_{\mathbf{w}\in S}\sum_{t=1}^T f(\mathbf{w};Z_t) ,$$
$$\mathrm{Diff}_T := \sum_{t=1}^T \left(F(\mathbf{w}_t) - F(\mathbf{w}^\star)\right) = \sum_{t=1}^T F(\mathbf{w}_t) - TF(\mathbf{w}^\star) .$$

Define the sequence of random variables
$$\xi_t := F(\mathbf{w}_t) - F(\mathbf{w}^\star) - \left(f(\mathbf{w}_t;Z_t) - f(\mathbf{w}^\star;Z_t)\right) . \tag{1}$$
Since $\mathbb{E}_{t-1}\left[f(\mathbf{w}_t;Z_t)\right] = F(\mathbf{w}_t)$ and $\mathbb{E}_{t-1}\left[f(\mathbf{w}^\star;Z_t)\right] = F(\mathbf{w}^\star)$, $\xi_t$ is a martingale difference sequence. This definition needs some explanation as it is important to look at the right martingale difference sequence to derive the results we want. Even under assumption LIST, $\frac{1}{T}\sum_t f(\mathbf{w}_t;Z_t)$ and $\frac{1}{T}\sum_t f(\mathbf{w}^\star;Z_t)$ will *not* be concentrated around $\frac{1}{T}\sum_t F(\mathbf{w}_t)$ and $F(\mathbf{w}^\star)$ respectively at a rate better then $O(1/\sqrt{T})$ in general. But if we look at the *difference*, we are able to get sharper concentration.

## 3   A General Online to Batch Conversion

The following simple lemma is crucial for us. It says that under assumption LIST, the variance of the increment in the regret $f(\mathbf{w}_t;Z_t) - f(\mathbf{w}^\star;Z_t)$ is bounded by its (conditional) expectation $F(\mathbf{w}_t) - F(\mathbf{w}^\star)$. Such a control on the variance is often the main ingredient in obtaining sharper concentration results.

**Lemma 1.** *Suppose assumption LIST holds and let $\xi_t$ be the martingale difference sequence defined in* (1). *Let*

$$\mathrm{Var}_{t-1}\xi_t := \mathbb{E}_{t-1}\left[\xi_t^2\right]$$

*be the conditional variance of $\xi_t$ given $Z_1, \ldots, Z_{t-1}$. Then, under assumption LIST, we have,*

$$\mathrm{Var}_{t-1}\xi_t \leq \frac{4L^2}{\nu}\left(F(\mathbf{w}_t) - F(\mathbf{w}^\star)\right) \ .$$

The variance bound given by the above lemma allows us to prove our main theorem.

**Theorem 2.** *Under assumption LIST, we have, with probability at least $1 - 4\ln(T)\delta$,*

$$\frac{1}{T}\sum_{t=1}^{T} F(\mathbf{w}_t) - F(\mathbf{w}^\star) \leq \frac{\mathrm{Reg}_T}{T} + 4\sqrt{\frac{L^2\ln(1/\delta)}{\nu}}\frac{\sqrt{\mathrm{Reg}_T}}{T} + \max\left\{\frac{16L^2}{\nu}, 6B\right\}\frac{\ln(1/\delta)}{T}$$

*Further, using Jensen's inequality, $\frac{1}{T}\sum_t F(\mathbf{w}_t)$ can be replaced by $F(\bar{\mathbf{w}})$ where $\bar{\mathbf{w}} := \frac{1}{T}\sum_t \mathbf{w}_t$.*

## 3.1 Proofs

*Proof of Lemma 1.* We have,

$$\mathrm{Var}_{t-1}\xi_t \leq \mathbb{E}_{t-1}\left[\left(f(\mathbf{w}_t; Z_t) - f(\mathbf{w}^\star; Z_t)\right)^2\right]$$

$$[\text{Assumption LIST, part 1}] \qquad \leq \mathbb{E}_{t-1}\left[L^2\|\mathbf{w}_t - \mathbf{w}^\star\|^2\right]$$

$$= L^2\|\mathbf{w}_t - \mathbf{w}^\star\|^2 \ . \tag{2}$$

On the other hand, using part 2 of assumption LIST, we also have for any $\mathbf{w}, \mathbf{w}' \in S$,

$$\frac{f(\mathbf{w}; Z) + f(\mathbf{w}'; Z)}{2} \geq f\left(\frac{\mathbf{w} + \mathbf{w}'}{2}; Z\right) + \frac{\nu}{8}\|\mathbf{w} - \mathbf{w}'\|^2 \ .$$

Taking expectation this gives, for any $\mathbf{w}, \mathbf{w}' \in S$,

$$\frac{F(\mathbf{w}) + F(\mathbf{w}')}{2} \geq F\left(\frac{\mathbf{w} + \mathbf{w}'}{2}\right) + \frac{\nu}{8}\|\mathbf{w} - \mathbf{w}'\|^2 \ .$$

Now using this with $\mathbf{w} = \mathbf{w}_t, \mathbf{w}' = \mathbf{w}^\star$, we get

$$\frac{F(\mathbf{w}_t) + F(\mathbf{w}^\star)}{2} \geq F\left(\frac{\mathbf{w}_t + \mathbf{w}^\star}{2}\right) + \frac{\nu}{8}\|\mathbf{w}_t - \mathbf{w}^\star\|^2$$

$$[\because \mathbf{w}^\star \text{ minimizes } F] \qquad \geq F(\mathbf{w}^\star) + \frac{\nu}{8}\|\mathbf{w}_t - \mathbf{w}^\star\|^2 \ .$$

This implies that

$$\|\mathbf{w}_t - \mathbf{w}^\star\|^2 \leq \frac{4(F(\mathbf{w}_t) - F(\mathbf{w}^\star))}{\nu} \tag{3}$$

Combining (2) and (3) we get,

$$\mathrm{Var}_{t-1}\xi_t \leq \frac{4L^2}{\nu}\left(F(\mathbf{w}_t) - F(\mathbf{w}^\star)\right)$$

$\square$

The proof of Theorem 2 relies on the following inequality for martingales which is an easy consequence of Freedman's inequality [Freedman, 1975, Theorem 1.6]. The proof of this lemma can be found in the appendix.

**Lemma 3.** *Suppose $X_1, \ldots, X_T$ is a martingale difference sequence with $|X_t| \leq b$. Let*

$$\mathrm{Var}_t X_t = \mathbf{Var}\left(X_t \mid X_1, \ldots, X_{t-1}\right) \ .$$

*Let $V = \sum_{t=1}^{T}\mathrm{Var}_t X_t$ be the sum of conditional variances of $X_t$'s. Further, let $\sigma = \sqrt{V}$. Then we have, for any $\delta < 1/e$ and $T \geq 3$,*

$$\mathrm{Prob}\left(\sum_{t=1}^{T} X_t > \max\left\{2\sigma, 3b\sqrt{\ln(1/\delta)}\right\}\sqrt{\ln(1/\delta)}\right) \leq 4\ln(T)\delta \ .$$

*Proof of Theorem 2.* By Lemma 1, we have $\sigma := \sqrt{\sum_{t=1}^{T} \mathrm{Var}_t \xi_t} \leq \sqrt{\frac{4L^2}{\nu} \mathrm{Diff}_T}$. Note that $|\xi_t| \leq 2B$ because our $f$ has range $[0, B]$. Therefore, Lemma 3 gives us that with probability at least $1 - 4\ln(T)\delta$, we have

$$\sum_{t=1}^{T} \xi_t \leq \max\left\{2\sigma, 6B\sqrt{\ln(1/\delta)}\right\} \sqrt{\ln(1/\delta)} .$$

By definition of $\mathrm{Reg}_T$,

$$\mathrm{Diff}_T - \mathrm{Reg}_T \leq \sum_{t=1}^{T} \xi_t$$

and therefore, with probability, $1 - 4\ln(T)\delta$, we have

$$\mathrm{Diff}_T - \mathrm{Reg}_T \leq \max\left\{4\sqrt{\frac{L^2}{\nu} \mathrm{Diff}_T}, 6B\sqrt{\ln(1/\delta)}\right\} \sqrt{\ln(1/\delta)} .$$

Using Lemma 4 below to solve the above quadratic inequality for $\mathrm{Diff}_T$, gives

$$\frac{\sum_{t=1}^{T} F(\mathbf{w}_t)}{T} - F(\mathbf{w}^\star) \leq \frac{\mathrm{Reg}_T}{T} + 4\sqrt{\frac{L^2 \ln(1/\delta)}{\nu}} \frac{\sqrt{\mathrm{Reg}_T}}{T} + \max\left\{\frac{16L^2}{\nu}, 6B\right\} \frac{\ln(1/\delta)}{T}$$

$\square$

The following elementary lemma was required to solve a recursive inequality in the proof of the above theorem. Its proof can be found in the appendix.

**Lemma 4.** *Suppose $s, r, d, b, \Delta \geq 0$ and we have*

$$s - r \leq \max\{4\sqrt{ds}, 6b\Delta\}\Delta .$$

*Then, it follows that*

$$s \leq r + 4\sqrt{dr}\Delta + \max\{16d, 6b\}\Delta^2 .$$

## 4 Applications

### 4.1 Online to Batch Conversion for Learning with Bounded Loss

Suppose $(X_1, Y_1), \ldots, (X_T, Y_T)$ are drawn i.i.d. from a distribution. The pairs $(X_i, Y_i)$ belong to $\mathcal{X} \times \mathcal{Y}$ and our algorithm are allowed to make predictions in a space $\mathcal{D} \supseteq \mathcal{Y}$. A loss function $\ell : \mathcal{D} \times \mathcal{Y} \to [0, 1]$ measures quality of predictions. Fix a convex set $S$ of some normed space and a function $h : \mathcal{X} \times S \to \mathcal{D}$. Let our hypotheses class be $\{x \mapsto h(x; \mathbf{w}) \mid \mathbf{w} \in S\}$.

On input $x$, the hypothesis parameterized by $\mathbf{w}$ predicts $h(x; \mathbf{w})$ and incurs loss $\ell(h(x; \mathbf{w}), y)$ if the correct prediction is $y$. The *risk* of $\mathbf{w}$ is defined by

$$R(\mathbf{w}) := \mathbb{E}\left[\ell(h(X; \mathbf{w}), Y)\right]$$

and let $\mathbf{w}^\star := \arg\min_{\mathbf{w} \in S} R(\mathbf{w})$ denote the (parameter for) the hypothesis with minimum risk. It is easy to see that this setting falls under the general framework given above by thinking of the pair $(X, Y)$ as $Z$ and setting $f(\mathbf{w}; Z) = f(\mathbf{w}; (X, Y))$ to be $\ell(h(X; \mathbf{w}), Y)$. Note that $F(\mathbf{w})$ becomes the risk $R(\mathbf{w})$. The range of $f$ is $[0, 1]$ by our assumption about the loss functions so $B = 1$.

Suppose we run an online algorithm on our data that generates a sequence of hypotheses $\mathbf{w}_0, \ldots, \mathbf{w}_T$ such that $\mathbf{w}_t$ is measurable w.r.t. $X_{<t}, Y_{<t}$. Define the statistics,

$$\mathrm{Reg}_T := \sum_{t=1}^{T} \ell(h(X_t; \mathbf{w}_t), Y_t) - \min_{\mathbf{w} \in S} \sum_{t=1}^{T} \ell(h(X_t; \mathbf{w}), Y_t) ,$$

$$\mathrm{Diff}_T := \sum_{t=1}^{T} (R(\mathbf{w}_t) - R(\mathbf{w}^\star)) = \sum_{t=1}^{T} R(\mathbf{w}_t) - T R(\mathbf{w}^\star) .$$

At the end, we output $\bar{\mathbf{w}} := (\sum_{t=1}^{T} \mathbf{w}_t)/T$. The following corollary then follows immediately from Theorem 2. It bounds the *excess risk* $R(\bar{\mathbf{w}}) - R(\mathbf{w}^\star)$.

**Corollary 5.** *Suppose assumption LIST is satisfied for $f(\mathbf{w}; (x, y)) := \ell(h(x; \mathbf{w}), y)$. Then we have, with probability at least $1 - 4\ln(T)\delta$,*

$$R(\bar{\mathbf{w}}) - R(\mathbf{w}^\star) \leq \frac{\text{Reg}_T}{T} + 4\sqrt{\frac{L^2 \ln(1/\delta)}{\nu}} \frac{\sqrt{\text{Reg}_T}}{T} + \max\left\{\frac{16L^2}{\nu}, 6\right\} \frac{\ln(1/\delta)}{T}$$

Recently, it has been proved [Kakade and Shalev-Shwartz, 2008] that if assumption LIST is satisfied for $\mathbf{w} \mapsto \ell(h(x; \mathbf{w}), y)$ then there is an online algorithm that generates $\mathbf{w}_1, \ldots, \mathbf{w}_T$ such that

$$\text{Reg}_T \leq \frac{L^2(1 + \ln T)}{2\nu} .$$

Plugging it in the corollary above gives the following result.

**Corollary 6.** *Suppose assumption LIST is satisfied for $f(\mathbf{w}; (x, y)) := \ell(h(x; \mathbf{w}), y)$. Then there is an online algorithm that generates $\mathbf{w}_1, \ldots, \mathbf{w}_T$ and in the end outputs $\bar{\mathbf{w}}$ such that, with probability at least $1 - 4\ln(T)\delta$,*

$$R(\bar{\mathbf{w}}) - R(\mathbf{w}^\star) \leq \frac{L^2 \ln T}{\nu T} + \frac{4L^2\sqrt{\ln T}}{\nu T}\sqrt{\ln\left(\frac{1}{\delta}\right)} + \max\left\{\frac{16L^2}{\nu}, 6\right\} \frac{\ln(1/\delta)}{T} ,$$

*for any $T \geq 3$.*

## 4.2 High Probability Bound for PEGASOS

PEGASOS [Shalev-Shwartz et al., 2007] is a recently proposed method for solving the primal SVM problem. Recall that in the SVM optimization problem we are given $m$ example, label pairs $(x_i, y_i) \in \mathbb{R}^d \times \{\pm 1\}$. Assume that $\|x_i\| \leq R$ for all $i$ where $\|\cdot\|$ is the standard $L_2$ norm. Let

$$F(\mathbf{w}) = \frac{\lambda}{2}\|\mathbf{w}\|^2 + \frac{1}{m}\sum_{i=1}^{m} \ell(\mathbf{w}; (x_i, y_i)) \tag{4}$$

be the SVM objective function. The loss function $\ell(\mathbf{w}; (x, y)) = [1 - y(\mathbf{w} \cdot x)]_+$ is the hinge loss. At time $t$, PEGASOS takes a (random) approximation

$$f(\mathbf{w}; Z_t) = \frac{\lambda}{2}\|\mathbf{w}\|^2 + \frac{1}{k}\sum_{(x,y) \in Z_t} \ell(\mathbf{w}; (x, y)) ,$$

of the SVM objective function to estimate the gradient and updates the current weight vector $\mathbf{w}_t$ to $\mathbf{w}_{t+1}$. Here $Z_t$ is a random subset of the data set of size $k$. Note that $F(\mathbf{w})$ can be written as

$$F(\mathbf{w}) = \mathbb{E}\left[\frac{\lambda^2}{2}\|\mathbf{w}\|^2 + \ell(\mathbf{w}; Z)\right]$$

where $Z$ is an example $(x_i, y_i)$ drawn uniformly at random from the $m$ data points. It is also easy to verify that

$$\forall \mathbf{w}, \ \mathbb{E}\left[f(\mathbf{w}; Z_t)\right] = F(\mathbf{w}) .$$

It can be shown that $\mathbf{w}^\star := \arg\min F(\mathbf{w})$ will satisfy $\|\mathbf{w}^\star\| \leq 1/\sqrt{\lambda}$ so we set

$$S = \left\{\mathbf{w} \in \mathbb{R}^d \ : \ \|\mathbf{w}\| \leq \frac{1}{\sqrt{\lambda}}\right\} .$$

For any $z$ that is a subset of the data set, the function

$$\mathbf{w} \mapsto f(\mathbf{w}; z) = \frac{\lambda}{2}\|\mathbf{w}\|^2 + \frac{1}{|z|}\sum_{(x,y) \in z} \ell(\mathbf{w}; (x, y))$$

is Lipschitz on $S$ with Lipschitz constant $L = \sqrt{\lambda} + R$ and is $\lambda$-strongly convex. Also $f(\mathbf{w}; z) \in [0, 3/2 + R/\sqrt{\lambda}]$. So, the PEGASOS setting falls under our general framework and satisfies assumption LIST.

Theorem 1 in Shalev-Shwartz et al. [2007] says, for any $\mathbf{w}$, $T \geq 3$,

$$\sum_{t=1}^{T} f(\mathbf{w}_t; Z_t) \leq \sum_{t=1}^{T} f(\mathbf{w}; Z_t) + \frac{L^2 \ln T}{\lambda} \, , \qquad (5)$$

where $L = \sqrt{\lambda} + R$. It was noted in that paper that plugging in $\mathbf{w} = \mathbf{w}^\star$ and taking expectations, we easily get

$$\mathbb{E}_{Z_1,\ldots,Z_T} \left[ \sum_{t=1}^{T} F(\mathbf{w}_t) \right] \leq T F(\mathbf{w}^\star) + \frac{L^2 \ln T}{\lambda} \, .$$

Here we use Theorem 2 to prove an inequality that holds with high probability, not just in expectation.

**Corollary 7.** *Let $F$ be the SVM objective function defined in (4) and $\mathbf{w}_1, \ldots, \mathbf{w}_T$ be the sequence of weight vectors generated by the* PEGASOS *algorithm. Further, let $\mathbf{w}^\star$ denote the minimizer of the SVM objective. Then, with probability $1 - 4\delta \ln(T)$, we have*

$$\sum_{t=1}^{T} F(\mathbf{w}_t) - T F(\mathbf{w}^\star) \leq \frac{L^2 \ln T}{\lambda} + \frac{4L^2 \sqrt{\ln T}}{\lambda} \sqrt{\ln\left(\frac{1}{\delta}\right)} + \max\left\{\frac{16L^2}{\lambda}, 9 + \frac{6R}{\sqrt{\lambda}}\right\} \ln\left(\frac{1}{\delta}\right) \, , \quad (6)$$

*for any $T \geq 3$. Therefore, assuming $R = 1$, we have, for $\lambda$ small enough, with probability at least $1 - \delta$,*

$$\frac{1}{T} \sum_{t=1}^{T} F(\mathbf{w}_t) - F(\mathbf{w}^\star) = O\left(\frac{\ln \frac{T}{\delta}}{\lambda T}\right) \, .$$

*Proof.* Note that (5) implies that $\text{Reg}_T \leq \frac{L^2 \ln T}{\lambda}$. The corollary then follows immediately from Theorem 2 by plugging in $\nu = \lambda$ and $B = 3/2 + R/\sqrt{\lambda}$. $\qquad \square$

## References

N. Cesa-Bianchi and C. Gentile. Improved risk tail bounds for on-line algorithms. *IEEE Transactions on Information Theory*, 54(1):286–390, 2008.

N. Cesa-Bianchi, A. Conconi, and C. Gentile. On the generalization ability of on-line learning algorithms. *IEEE Transactions on Information Theory*, 50(9):2050–2057, September 2004.

M. Collins. Discriminative training methods for hidden Markov models: Theory and experiments with perceptron algorithms. In *Conference on Empirical Methods in Natural Language Processing*, 2002.

K. Crammer, O. Dekel, J. Keshet, S. Shalev-Shwartz, and Y. Singer. Online passive aggressive algorithms. *Journal of Machine Learning Research*, 7:551–585, Mar 2006.

David A. Freedman. On tail probabilities for martingales. *The Annals of Probability*, 3(1):100–118, Feb 1975.

E. Hazan, A. Kalai, S. Kale, and A. Agarwal. Logarithmic regret algorithms for online convex optimization. In *Proceedings of the Nineteenth Annual Conference on Computational Learning Theory*, 2006.

S. Kakade and S. Shalev-Shwartz. Mind the duality gap: Logarithmic regret algorithms for online optimization. *Advances in Neural Information Processing Systems*, 2008.

N. Littlestone. *Mistake bounds and logarithmic linear-threshold learning algorithms.* PhD thesis, U. C. Santa Cruz, March 1989.

Nathan Ratliff, James (Drew) Bagnell, and Martin Zinkevich. (online) subgradient methods for structured prediction. In *Eleventh International Conference on Artificial Intelligence and Statistics (AIStats)*, March 2007.

Shai Shalev-Shwartz, Yoram Singer, and Nathan Srebro. Pegasos: Primal Estimated sub-GrAdient SOlver for SVM. In *Proceedings of the Twenty-Fourth International Conference on Machine Learning (ICML)*, pages 807–814, 2007.

T. Zhang. Data dependent concentration bounds for sequential prediction algorithms. In *Proceedings of the Eighteenth Annual Conference on Computational Learning Theory*, pages 173–187, 2005.

## Appendix

*Proof of Lemma 3.* Note that a crude upper bound on $\mathrm{Var}_t X_t$ is $b^2$. Thus, $\sigma \le b\sqrt{T}$. We choose a discretization $0 = \alpha_{-1} < \alpha_0 < \ldots < \alpha_l$ such that $\alpha_{i+1} = r\alpha_i$ for $i \ge 0$ and $\alpha_l \ge b\sqrt{T}$. We will specify the choice of $\alpha_0$ and $r$ shortly. We then have, for any $c > 0$,

$$
\mathrm{Prob}\left(\sum_t X_t > c \max\{r\sigma, \alpha_0\}\sqrt{\ln(1/\delta)}\right)
$$

$$
= \sum_{j=0}^{l} \mathrm{Prob}\left(\begin{array}{c}\sum_t X_t > c\max\{r\sigma, \alpha_0\}\sqrt{\ln(1/\delta)} \\ \& \ \alpha_{j-1} < \sigma \le \alpha_j\end{array}\right)
$$

$$
\le \sum_{j=0}^{l} \mathrm{Prob}\left(\begin{array}{c}\sum_t X_t > c\alpha_j\sqrt{\ln(1/\delta)} \\ \& \ \alpha_{j-1}^2 < V \le \alpha_j^2\end{array}\right)
$$

$$
\le \sum_{j=0}^{l} \mathrm{Prob}\left(\sum_t X_t > c\alpha_j\sqrt{\ln(1/\delta)} \ \& \ V \le \alpha_j^2\right)
$$

$$
\overset{(\star)}{\le} \sum_{j=0}^{l} \exp\left(\frac{-c^2\alpha_j^2\ln(1/\delta)}{2\alpha_j^2 + \frac{2}{3}\left(c\alpha_j\sqrt{\ln(1/\delta)}\right)b}\right)
$$

$$
= \sum_{j=0}^{l} \exp\left(\frac{-c^2\alpha_j\ln(1/\delta)}{2\alpha_j + \frac{2}{3}\left(c\sqrt{\ln(1/\delta)}\right)b}\right)
$$

where the inequality $(\star)$ follows from Freedman's inequality. If we now choose $\alpha_0 = bc\sqrt{\ln(1/\delta)}$ then $\alpha_j \ge bc\sqrt{\ln(1/\delta)}$ for all $j$ and hence every term in the above summation is bounded by $\exp\left(\frac{-c^2\ln(1/\delta)}{2+2/3}\right)$ which is less then $\delta$ if we choose $c = 5/3$. Set $r = 2/c = 6/5$. We want $\alpha_0 r^l \ge b\sqrt{T}$. Since $c\sqrt{\ln(1/\delta)} \ge 1$, choosing $l = \log_r(\sqrt{T})$ ensures that. Thus we have

$$
\mathrm{Prob}\left(\sum_{t=1}^{T} X_t > \frac{5}{3}\max\{\frac{6}{5}\sigma, \frac{5}{3}b\sqrt{\ln(1/\delta)}\}\sqrt{\ln(1/\delta)}\right)
$$

$$
= \mathrm{Prob}\left(\sum_t X_t > c\max\{r\sigma, \alpha_0\}\sqrt{\ln(1/\delta)}\right)
$$

$$
\le (l+1)\delta = (\log_{6/5}(\sqrt{T}) + 1)\delta
$$

$$
\le (6\ln(\sqrt{T}) + 1)\delta \le 4\ln(T)\delta . \qquad\qquad (\because T \ge 3)
$$

$\square$

*Proof of Lemma 4.* The assumption of the lemma implies that one of the following inequalities holds:

$$
s - r \le 6b\Delta^2 \qquad\qquad\qquad s - r \le 4\sqrt{ds}\Delta . \qquad\qquad (7)
$$

In the second case, we have

$$
\left(\sqrt{s}\right)^2 - (4\sqrt{d}\Delta)\sqrt{s} - r \le 0
$$

which means that $\sqrt{s}$ should be smaller than the larger root of the above quadratic. This gives us,

$$
s = (\sqrt{s})^2 \le \left(2\sqrt{d}\Delta + \sqrt{4d\Delta^2 + r}\right)^2
$$

$$
\le 4d\Delta^2 + 4d\Delta^2 + r + 4\sqrt{4d^2\Delta^4 + d\Delta^2 r}
$$

$$
[\because \sqrt{x+y} \le \sqrt{x} + \sqrt{y}] \qquad \le 8d\Delta^2 + r + 8d\Delta^2 + 4\sqrt{dr}\Delta
$$

$$
\le r + 4\sqrt{dr}\Delta + 16d\Delta^2 . \qquad\qquad (8)
$$

Combining (7) and (8) finishes the proof. $\square$